# Discriminant Adaptive Nearest Neighbor Classification and Regression

**Trevor Hastie**
Department of Statistics
Sequoia Hall
Stanford University
California 94305
trevor@playfair.stanford.edu

**Robert Tibshirani**
Department of Statistics
University of Toronto
tibs@utstat.toronto.edu

## Abstract

Nearest neighbor classification expects the class conditional probabilities to be locally constant, and suffers from bias in high dimensions We propose a locally adaptive form of nearest neighbor classification to try to finesse this curse of dimensionality. We use a local linear discriminant analysis to estimate an effective metric for computing neighborhoods. We determine the local decision boundaries from centroid information, and then shrink neighborhoods in directions orthogonal to these local decision boundaries, and elongate them parallel to the boundaries. Thereafter, any neighborhood-based classifier can be employed, using the modified neighborhoods. We also propose a method for global dimension reduction, that combines local dimension information. We indicate how these techniques can be extended to the regression problem.

## 1  Introduction

We consider a discrimination problem with $J$ classes and $N$ training observations. The training observations consist of predictor measurements $\mathbf{x} = (x_1, x_2, \ldots x_p)$ on $p$ predictors and the known class memberships. Our goal is to predict the class membership of an observation with predictor vector $\mathbf{x}_0$

Nearest neighbor classification is a simple and appealing approach to this problem. We find the set of $K$ nearest neighbors in the training set to $\mathbf{x}_0$ and then classify $\mathbf{x}_0$ as the most frequent class among the $K$ neighbors.

Cover & Hart (1967) show that the one nearest neighbour rule has asymptotic error rate at most twice the Bayes rate. However in finite samples the curse of

dimensionality can severely hurt the nearest neighbor rule. The relative radius of the nearest-neighbor sphere grows like $r^{1/p}$ where $p$ is the dimension and $r$ the radius for $p = 1$, resulting in severe bias at the target point $\mathbf{x}$. Figure 1 (left panel) illustrates the situation for a simple example. Nearest neighbor techniques are

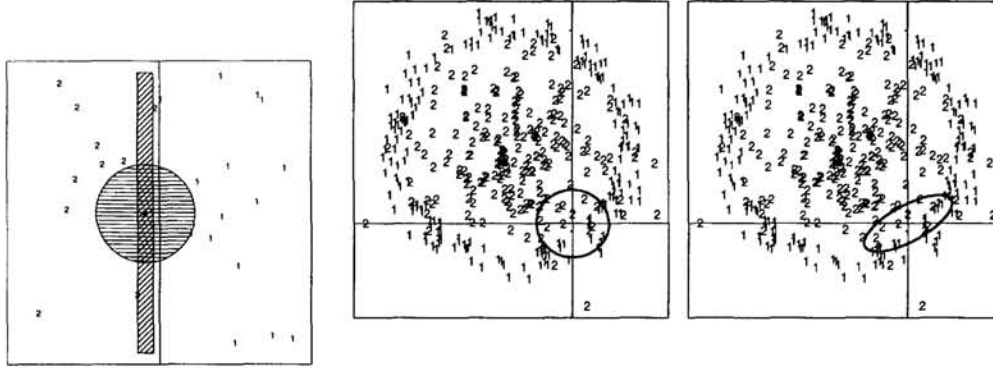

Figure 1: *In the left panel, the vertical strip denotes the NN region using only horizontal coordinate to find the nearest neighbor for the target point (solid dot). The sphere shows the NN region using both coordinates, and we see in this case it has extended into the class 1 region (and found the wrong class in this instance). The middle panel shows a spherical neighborhood containing 25 points, for a two class problem with a circular decision boundary. The right panel shows the ellipsoidal neighborhood found by the* DANN *procedure, also containing 25 points. The latter is elongated in a direction parallel to the true decision boundary (locally constant posterior probabilities), and flattened orthogonal to it.*

based on the assumption that locally the class posterior probabilities are constant. While that is clearly true in the vertical strip using only the vertical coordinate, using both this is no longer true. Figure 1 (middle and right panels) shows how we locally adapt the metric to overcome this problem, in a situation where the decision boundary is locally linear.

## 2   Discriminant adaptive nearest neighbors

Consider first a standard linear discriminant (LDA) classification procedure with $K$ classes. Let $\mathbf{B}$ and $\mathbf{W}$ denote the between and within sum of squares matrices. In LDA the data are first sphered with respect to $\mathbf{W}$, then the target point is classified to the class of the closest centroid (with a correction for the class prior membership probabilities). Since only relative distances are relevant, any distances in the complement of the subspace spanned by the sphered centroids can be ignored. This complement corresponds to the null space of $\mathbf{B}$.

We propose to estimate $\mathbf{B}$ and $\mathbf{W}$ locally, and use them to form a local metric that approximately behaves like the LDA metric. One such candidate is

$$
\begin{aligned}
\Sigma &= \mathbf{W}^{-1}\mathbf{B}\mathbf{W}^{-1} \\
&= \mathbf{W}^{-1/2}(\mathbf{W}^{-1/2}\mathbf{B}\mathbf{W}^{-1/2})\mathbf{W}^{-1/2} \\
&= \mathbf{W}^{-1/2}\mathbf{B}^*\mathbf{W}^{-1/2}.
\end{aligned}
\tag{1}
$$

where $\mathbf{B}^*$ is the between sum-of-squares in the sphered space. Consider the action of $\Sigma$ as a metric for computing distances

$$
(\mathbf{x} - \mathbf{x}_0)^T \Sigma (\mathbf{x} - \mathbf{x}_0) :
\tag{2}
$$

- it first spheres the space using $\mathbf{W}$;

- components of distance in the null space of $\mathbf{B}^*$ are ignored;

- other components are weighted according to the eigenvalues of $\mathbf{B}^*$ when there are more than 2 classes — directions in which the centroids are more spread out are weighted more than those in which they are close

Thus this metric would result in neighborhoods similar to the narrow strip in figure 1(left figure): infinitely long in the null space of $\mathbf{B}$, and then deformed appropriately in the centroid subspace according to how they are placed. It is dangerous to allow neighborhoods to extend infinitely in any direction, so we need to limit this stretching. Our proposal is

$$
\begin{aligned}
\Sigma &= \mathbf{W}^{-1/2}[\mathbf{W}^{-1/2}\mathbf{B}\mathbf{W}^{-1/2} + \epsilon\mathbf{I}]\mathbf{W}^{-1/2} \\
&= \mathbf{W}^{-1/2}[\mathbf{B}^* + \epsilon\mathbf{I}]\mathbf{W}^{-1/2}
\end{aligned} \tag{3}
$$

where $\epsilon$ is some small tuning parameter to be determined. The metric shrinks the neighborhood in directions in which the local class centroids differ, with the intention of ending up with a neighborhood in which the class centroids coincide (and hence nearest neighbor classification is appropriate). Given $\Sigma$ we use perform $K$-nearest neighbor classification using the metric (2).

There are several details that we briefly describe here and in more detail in Hastie & Tibshirani (1994):

- $\mathbf{B}$ is defined to be the covariance of the class centroids, and $\mathbf{W}$ the pooled estimate of the common class covariance matrix. We estimate these locally using a *spherical*, compactly supported kernel (Cleveland 1979), where the bandwidth is determined by the distance of the $K_M$ nearest neighbor.

- $K_M$ above has to be supplied, as does the softening parameter $\epsilon$. We somewhat arbitrarily use $K_M = \max(N/5, 50)$; so we use many more neighbors (50 or more) to determine the metric, and then typically $K = 1, \ldots, 5$ nearest neighbors in this metric to classify. We have found that the metric is relatively insensitive to different values of $0 < \epsilon < 5$, and typically use $\epsilon = 1$.

- Typically the data do not support the local calculation of $\mathbf{W}$ ($p(p+1)/2$ entries), and it can be argued that this is not necessary. We mostly resort to using the diagonal of $\mathbf{W}$ instead, or else use a global estimate.

Sections 4 and 5 illustrate the effectiveness of this approach on some simulated and real examples.

## 3   Dimension Reduction using Local Discriminant Information

The technique described above is entirely "memory based", in that we locally adapt a neighborhood about a query point at the time of classification. Here we describe a method for performing a global dimension reduction, by pooling the local dimension information over all points in the training set. In a nutshell we consider subspaces corresponding to *eigenvectors of the average local between sum-of-squares matrices*.

Consider first how linear discriminant analysis (LDA) works. After sphering the data, it concentrates in the space spanned by the class centroids $\bar{\mathbf{x}}_j$ or a reduced rank space that lies close to these centroids. If $\bar{\mathbf{x}}$ denote the overall centroid, this

subspace is exactly a principal component hyperplane for the data points $\bar{\mathbf{x}}_j - \bar{\mathbf{x}}$, weighted by the class proportions, and is given by the eigen-decomposition of the between covariance $\mathbf{B}$.

Our idea to compute the deviations $\bar{\mathbf{x}}_j - \bar{\mathbf{x}}$ locally in a neighborhood around each of the $N$ training points, and then do an overall principal components analysis for the $N \times J$ deviations. This amounts to an eigen-decomposition of the average between sum of squares matrix $\sum_{i=1}^{N} \mathbf{B}(i)/N$.

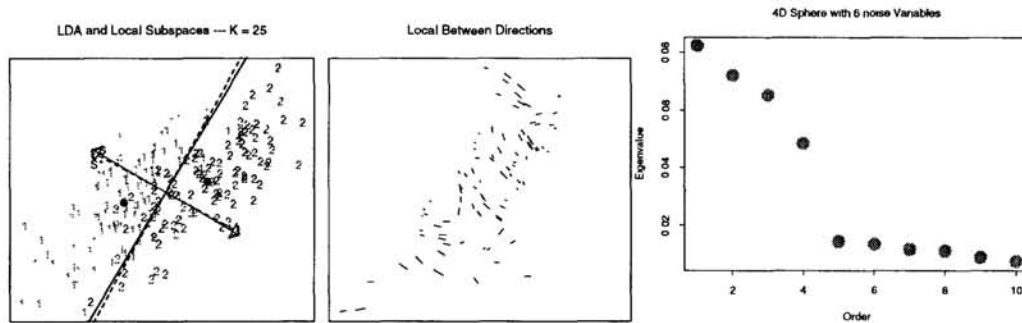

Figure 2: *[Left Panel] Two dimensional gaussian data with two classes and correlation 0.65. The solid lines are the LDA decision boundary and its equivalent subspace for classification, computed using both the between and (crucially) the within class covariance. The dashed lines were produced by the local procedure described in this section, without knowledge of the overall within covariance matrix. [Middle panel] Each line segment represents the local between information centered at that point. [Right panel] The eigenvalues of the average between matrix for the 4D sphere in 10D problem. Using these first four dimensions followed by our* **DANN** *nearest neighbor routine, we get better performance than 5NN in the real 4D subspace.*

Figure 2 (left two panels) demonstrates by a simple illustrative example that our subspace procedure can recover the correct LDA direction without making use of the within covariance matrix. Figure 2 (right panel) represents a two class problem with a 4-dimensional spherical decision boundary. The data for the two classes lie in concentric spheres in 4D, the one class lying inside the other with some overlap (a 4D version of the same 2D situation in figure 1.) In addition the are an extra 6 noise dimensions, and for future reference we denote such a model as the "4D spheres in 10D" problem. The decision boundary is a 4 dimensional sphere, although locally linear. The eigenvalues show a distinct change after 4 (the correct dimension), and using our **DANN** classifier in these four dimensions actually beats ordinary 5NN in the *known* 4D discriminant subspace.

## 4    Examples

Figure 3 summarizes the results of a number of simulated examples designed to test our procedures in both favorable and unfavorable situations. In all the situations **DANN** outperforms 5-NN. In the cases where 5NN is provided with the known lower-dimensional discriminant subspace, our subspace technique **subDANN** followed by **DANN** comes close to the optimal performance.

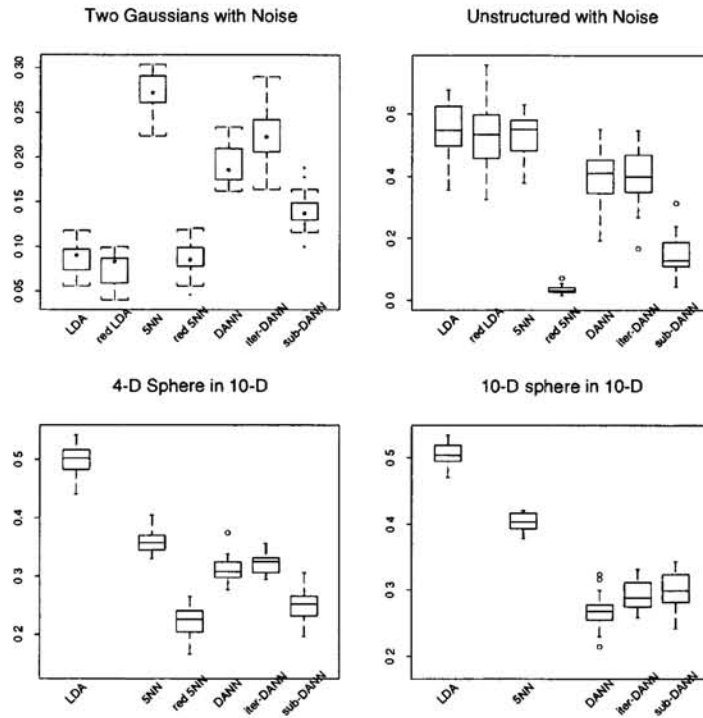

Figure 3: *Boxplots of error rates over 20 simulations. The top left panel has two gaussian distributions separated in two dimensions, with 14 noise dimensions. The notation* red-LDA *and* red-5NN *refers to these procedures in the known lower dimensional space.* iter-DANN *refers to an iterated version of* DANN *(which appears not to help), while* sub-DANN *refers to our global subspace approach, followed by* DANN*. The top right panel has 4 classes, each of which is a mixture of 3-gaussians in 2-D; in addition there are 8 noise variables. The lower two panels are versions of our sphere example.*

## 5   Image Classification Example

Here we consider an image classification problem. The data consist of 4 LANDSAT images in different spectral bands of a small area of the earths surface, and the goal is to classify into soil and vegetation types. Figure 4 shows the four spectral bands, two in the visible spectrum (red and green) and two in the infra red spectrum. These data are taken from the data archive of the STATLOG (Michie et al. 1994)[1]. The goal is to classify each pixel into one of 7 land types: *red soil, cotton, vegetation stubble, mixture, grey soil, damp grey soil, very damp grey soil.* We extract for each pixel its 8-neighbors, giving us $(8 + 1) \times 4 = 36$ features (the pixel intensities) per pixel to be classified. The data come scrambled, with 4435 training pixels and 2000 test pixels, each with their 36 features and the known classification. Included in figure 4 is the true classification, as well as that produced by linear discriminant analysis. The right panel compares DANN to all the procedures used in STATLOG, and we see the results are favorable.

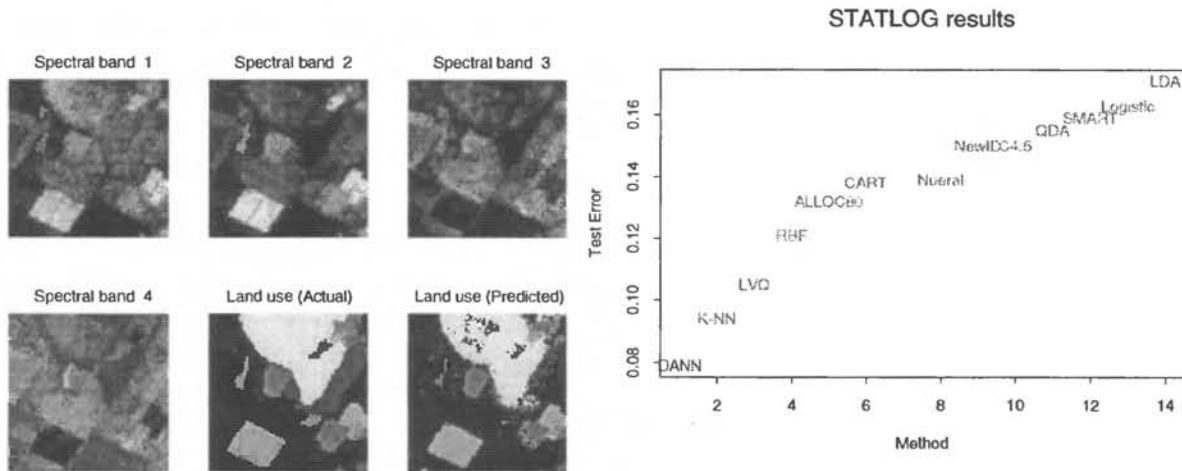

Figure 4: The first four images are the satellite images in the four spectral bands. The fifth image represents the known classification, and the final image is the classification map produced by linear discriminant analysis. The right panel shows the misclassification results of a variety of classification procedures on the satellite image test data (taken from Michie et al. (1994)). DANN is the overall winner.

## 6    Local Regression

Near neighbor techniques are used in the regression setting as well. Local polynomial regression (Cleveland 1979) is currently very popular, where, for example, locally weighted linear surfaces are fit in modest sized neighborhoods. Analogs of K-NN classification for small $K$ are used less frequently. In this case the response variable is quantitative rather than a class label.

Duan & Li (1991) invented a technique called *sliced inverse regression*, a dimension reduction tool for situations where the regression function changes in a lower-dimensional space. They show that under symmetry conditions of the marginal distribution of $X$, the inverse regression curve $E(X|Y)$ is concentrated in the same lower-dimensional subspace. They estimate the curve by slicing $Y$ into intervals, and computing conditional means of $X$ in each interval, followed by a principal component analysis. There are obvious similarities with our DANN procedure, and the following generalizations of DANN are suggested for regression:

- locally we use the **B** matrix of the sliced means to form our DANN metric, and then perform local regression in the deformed neighborhoods.
- The local **B**($i$) matrices can be pooled as in subDANN to extract global subspaces for regression. This has an apparent advantage over the Duan & Li (1991) approach: we only require symmetry locally, a condition that is locally encouraged by the convolution of the data with a spherical kernel[2]

## 7    Discussion

Short & Fukanaga (1980) proposed a technique close to ours for the two class problem. In our terminology they used our metric with $\mathbf{W} = \mathbf{I}$ and $\epsilon = 0$, with **B** determined locally in a neighborhood of size $K_M$. In effect this extends the

neighborhood infinitely in the null space of the local between class directions, but they restrict this neighborhood to the original $K_M$ observations. This amounts to projecting the local data onto the line joining the two local centroids. In our experiments this approach tended to perform on average 10% worse than our metric, and we did not pursue it further. Short & Fukanaga (1981) extended this to $J > 2$ classes, but here their approach differs even more from ours. They computed a weighted average of the $J$ local centroids from the overall average, and project the data onto it, a one dimensional projection. Myles & Hand (1990) recognized a shortfall of the Short and Fukanaga approach, since the averaging can cause cancellation, and proposed other metrics to avoid this, different from ours.

Friedman (1994) proposes a number of techniques for flexible metric nearest neighbor classification (and sparked our interest in the problem.) These techniques use a recursive partitioning style strategy to adaptively shrink and shape rectangular neighborhoods around the test point.

## Acknowledgement

The authors thank Jerry Friedman whose research on this problem was a source of inspiration, and for many discussions. Trevor Hastie was supported by NSF DMS-9504495. Robert Tibshirani was supported by a Guggenheim fellowship, and a grant from the National Research Council of Canada.

## Footnotes

[1]The authors thank C. Taylor and D. Spiegelhalter for making these images and data available

[2]We expect to be able to substantiate the claims in this section by the time of the NIPS995 meeting.

## References

Cleveland, W. (1979), 'Robust locally-weighted regression and smoothing scatterplots', *Journal of the American Statistical Society* **74**, 829–836.

Cover, T. & Hart, P. (1967), 'Nearest neighbor pattern classification', *Proc. IEEE Trans. Inform. Theory* pp. 21–27.

Duan, N. & Li, K.-C. (1991), 'Slicing regression: a link-free regression method', *Annals of Statistics* pp. 505–530.

Friedman, J. (1994), Flexible metric nearest neighbour classification, Technical report, Stanford University.

Hastie, T. & Tibshirani, R. (1994), Discriminant adaptive nearest neighbor classification, Technical report, Statistics Department, Stanford University.

Michie, D., Spigelhalter, D. & Taylor, C., eds (1994), *Machine Learning, Neural and Statistical Classification*, Ellis Horwood series in Artificial Intelligence, Ellis Horwood.

Myles, J. & Hand, D. J. (1990), 'The multi-class metric problem in nearest neighbour discrimination rules', *Pattern Recognition* **23**, 1291–1297.

Short, R. & Fukanaga, K. (1980), A new nearest neighbor distance measure, *in* 'Proc. 5th IEEE Int. Conf. on Pattern Recognition', pp. 81–86.

Short, R. & Fukanaga, K. (1981), 'The optimal distance measure for nearest neighbor classification', *IEEE transactions of Information Theory* **IT-27**, 622–627.
